# A Principle for Unsupervised Hierarchical Decomposition of Visual Scenes

**Michael C. Mozer**
*Dept. of Computer Science*
*University of Colorado*
*Boulder, CO 80309–0430*

## ABSTRACT

Structure in a visual scene can be described at many levels of granularity. At a coarse level, the scene is composed of objects; at a finer level, each object is made up of parts, and the parts of subparts. In this work, I propose a simple principle by which such hierarchical structure can be extracted from visual scenes: Regularity in the relations among different parts of an object is weaker than in the internal structure of a part. This principle can be applied recursively to define part-whole relationships among elements in a scene. The principle does not make use of object models, categories, or other sorts of higher-level knowledge; rather, part-whole relationships can be established based on the statistics of a set of sample visual scenes. I illustrate with a model that performs unsupervised decomposition of simple scenes. The model can account for the results from a human learning experiment on the ontogeny of part-whole relationships.

## 1 INTRODUCTION

The structure in a visual scene can be described at many levels of granularity. Consider the scene in Figure 1a. At a coarse level, the scene might be said to consist of stick man and stick dog. However, stick man and stick dog themselves can be decomposed further. One might describe stick man as having two components, a head and a body. The head in turn can be described in terms of its parts: the eyes, nose, and mouth. This sort of scene decomposition can continue recursively down to the level of the primitive visual features. Figure 1b shows a partial decomposition of the scene in Figure 1a.

A scene decomposition establishes part-whole relationships among objects. For example, the mouth (a whole) consists of two parts, the teeth and the lips. If we assume that any part can belong to only one whole, the decomposition imposes a *hierarchical* structure over the elements in the scene.

Where does this structure come from? What makes an object an object, a part a part? I propose a simple principle by which such hierarchical structure can be extracted from visual scenes and incorporate the principle in a simulation model. The principle is based on the statistics of the visual environment, not on object models or other sorts of higher-level knowledge, or on a teacher to classify objects or their parts.

## 2  WHAT MAKES A PART A PART?

Parts combine to form objects. Parts are combined in different ways to form different objects and different instances of an object. Consequently, the structural relations among different parts of an object are less *regular* than is the internal structure of a part. To illustrate, consider Figure 2, which depicts four instances of a box shell and lid. The components of the lid—the top and the handle—appear in a regular configuration, as do the components of the shell—the sides and base—but the relation of the lid to the shell is variable. Thus, configural regularity is an indication that components should be grouped together to form a unit. I call this the *regularity principle*. Other variants of the regularity principle have been suggested by Becker (1995) and Tenenbaum (1994).

The regularity depicted in Figure 2 is quite rigid: one component of a part always occurs in a fixed spatial position relative to another. The regularity principle can also be cast in terms of abstract relationships such as containment and encirclement. The only difference is the featural representation that subserves the regularity discovery process. In this paper, however, I address primarily regularities that are based on physical features and fixed spatial relationships. Another generalization of the regularity principle is that it can be applied recursively to suggest not only parts of wholes, but subparts of parts.

According to the regularity principle, information is implicit in the environment that can be used to establish part-whole relationships. This information comes in the form of statistical regularities among features in a visual scene. The regularity principle does not depend on explicit labeling of parts or objects.

In contrast, Schyns and Murphy (1992, 1993) have suggested a theory of part ontogeny that presupposes explicit categorization of objects. They propose a *homogeneity principle* which states that "if a fragment of a stimulus plays a consistent role in categorization, the perceptual parts composing the fragment are instantiated as a single unit in the stimulus representation in memory." Their empirical studies with human subjects find support for the homogeneity principle.

Superficially, the homogeneity and regularity principles seem quite different: while the homogeneity principle applies to *supervised* category learning (i.e., with a teacher to classify instances), the regularity principle applies to *unsupervised* discovery. But it is possible to transform one learning paradigm into the other. For example, in a category learning task, if only one category is to be learned and if the training examples are all positive instances of the category, then inducing the defining characteristics of the category is equivalent to extracting regularities in the stimulus environment. Thus, category learning in a diverse stimulus environment can be conceptualized as unsupervised regularity extraction in multiple, narrow stimulus environments (each environment being formed by taking all positive instances of a given class).

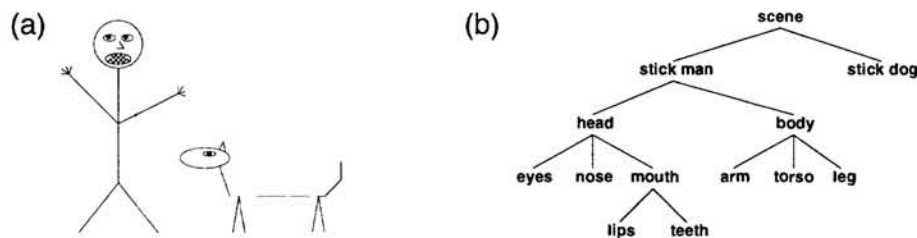

**FIGURE 1. (a) A graphical depiction of stick man and his faithful companion, stick dog; (b) a partial decomposition of the scene into its parts.**

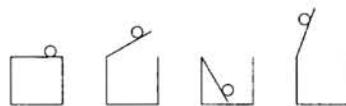

**FIGURE 2. Four different instances of a box with a lid**

There are several other differences between the regularity principle proposed here and the homogeneity principle of Schyns and Murphy, but they are minor. Schyns and Murphy seem to interpret "fragment" more narrowly as spatially contiguous perceptual features. They also don't address the hierarchical nature of part-whole relationships. Nonetheless, the two principles share the notion of using the statistical structure of the visual environment to establish part-whole relations.

## 3   A FLAT REPRESENTATION OF STRUCTURE

I have incorporated the regularity principle into a neural net that discovers part-whole relations in its environment. Neural nets, having powerful learning paradigms for unsupervised discovery, are well suited for this task. However, they have a fundamental difficulty representing complex, articulated data structures of the sort necessary to encode hierarchies (but see Pollack, 1988, and Smolensky, 1990, for promising advances). I thus begin by describing a novel representation scheme for hierarchical structures that can readily be integrated into a neural net.

The tree structure in Figure 1b depicts one representation of a hierarchical decomposition. The complete tree has as its leaf nodes the primitive visual features of the scene. The tree specifies the relationships among the visual features. There is another way of capturing these relationships, more connectionist in spirit than the tree structure. The idea is to assign to each primitive feature a *tag*—a scalar in [0, 1]—such that features within a subtree have similar values. For the features of stick man, possible tags might be: **eyes** .1, **nose** .2, **lips** .28, **teeth** .32, **arm** .6, **torso** .7, **leg** .8.

Denoting the set of all features having tags in $[\alpha, \beta]$ by $S(\alpha, \beta)$, one can specify any subtree of the stick man representation. For example, $S(0,1)$ includes all features of stick man; $S(0,.5)$ includes all features in the subtree whose root is stick man's head, $S(.5,1)$ his body; $S(.25,.35)$ indicates the parts of the mouth. By a simple algorithm, tags can be assigned to the leaf nodes of any tree such that any subtree can be selected by specifying an appropriate tag range. The only requirement for this algorithm is knowledge of the maximum branching factor. There is no fixed limit to the depth of the tree that can be thus represented; however, the deeper the tree, the finer the tag resolution that will be needed.

The tags provide a "flat" way of representing hierarchical structure. Although the tree is implicit in the representation, the tags convey all information in the tree, and thus can capture complex, articulated structures. The tags in fact convey additional information. For example in the above feature list, note that **lips** is closer to **nose** than **teeth** is to **nose**. This information can easily be ignored, but it is still worth observing that the tags carry extra baggage not present in the symbolic tree structure.

It is convenient to represent the tags on a range $[0, 2\pi)$ rather than $[0,1]$. This allows the tag to be identified with a directional—or angular—value. Viewed as part of a cyclic continuum, the directional tags are homogeneous, in contrast to the linear tags where tags near 0 and 1 have special status by virtue of being at endpoints of the continuum. Homogeneity results in a more elegant model, as described below.

The directional tags also permit a neurophysiological interpretation, albeit speculative. It has been suggested that synchronized oscillatory activities in the nervous system can be used to convey information above and beyond that contained in the average firing rate of individual neurons (e.g., Eckhorn et al., 1988; Gray et al., 1989; von der Malsburg, 1981). These oscillations vary in their *phase*, the relative offset of the bursts. The directional tags could map directly to phases of oscillations, providing a means of implementing the tagging in neocortex.

## 4   REGULARITY DISCOVERY

Many learning paradigms allow for the discovery of regularity. I have used an autoencoder architecture (Plaut, Nowlan, & Hinton, 1986) that maps an input pattern—a

representation of visual features in a scene—to an output pattern via a small layer of hidden units. The goal of this type of architecture is for the network to reproduce the input pattern over the output units. The task requires discovery of regularities because the hidden layer serves as an encoding bottleneck that limits the representational capacity of the system. Consequently, stronger regularities (the most common patterns) will be encoded over the weaker.

## 5 MAGIC

We now need to combine the autoencoder architecture with the notion of tags such that regularity of feature configurations in the input will increase the likelihood that the features will be assigned the same tags.

This goal can be achieved using a model we developed for segmenting an image into different objects using supervised learning. The model, *MAGIC* (Mozer, Zemel, Behrmann, & Williams, 1992), was trained on images containing several visual objects and its task was to tag features according to which object they belonged. A teacher provided the target tags. Each unit in MAGIC conveys two distinct values: a probability that a feature is present, which I will call the feature *activity*, and a tag associated with the feature. The tag is a directional (angular) value, of the sort suggested earlier. (The tag representation is in reality a complex number whose direction corresponds to the directional value and whose magnitude is related to the unit's confidence in the direction. As this latter aspect of the representation is not central to the present work, I discuss it no further.)

The architecture is a two layer recurrent net. The input or *feature* layer is set of spatiotopic arrays—in most simulations having dimensions 25×25—each array containing detectors for features of a given type: oriented line segments at 0°, 45°, 90°, and 135°. In addition, there is a layer of *hidden* units. Each hidden unit is reciprocally connected to input from a local spatial *patch* of the input array; in the current simulations, the patch has dimensions 4×4. For each patch there is a corresponding fixed-size *pool* of hidden units. To achieve a translation invariant response across the image, the pools are arranged in a spatiotopic array in which neighboring pools respond to neighboring patches and the patch-to-pool weights are constrained to be the same at all locations in the array. There are interlayer connections, but no intralayer connections.

The images presented to MAGIC consist of an arrangement of features over the input array. The feature activity is clamped on (i.e., the feature is present), and the initial directional tag of the feature is set at random. Feature unit activities and tags feed to the hidden units, which in turn feed back to the feature units. Through a relaxation process, the system settles on an assignment of tags to the feature units (as well as to the hidden units, although read out from the model concerns only the feature units). MAGIC is a mean-field approximation to a stochastic network of directional units with binary-gated outputs (Zemel, Williams, & Mozer, 1995). This means that a mean-field energy functional can be written that expresses the network state and controls the dynamics; consequently, MAGIC is guaranteed to converge to a stable pattern of tags.

Each hidden unit detects a spatially local configuration of features, and it acts to reinstate a pattern of tags over the configuration. By adjusting its incoming and outgoing weights during training, the hidden unit is made to respond to configurations that are consistently tagged in the training set. For example, if the training set contains many corner junctions where horizontal and vertical lines come to a point and if the teacher tags all features composing these lines as belonging to the same object, then a hidden unit might learn to detect this configuration, and when it does so, to force the tags of the component features to be the same.

In our earlier work, MAGIC was trained to map the feature activity pattern to a target pattern of feature tags, where there was a distinct tag for each object in the image. In the present work, the training objective is rather to impose *uniform* tags over the features. Additionally, the training objective encourages MAGIC to reinstate the feature activity

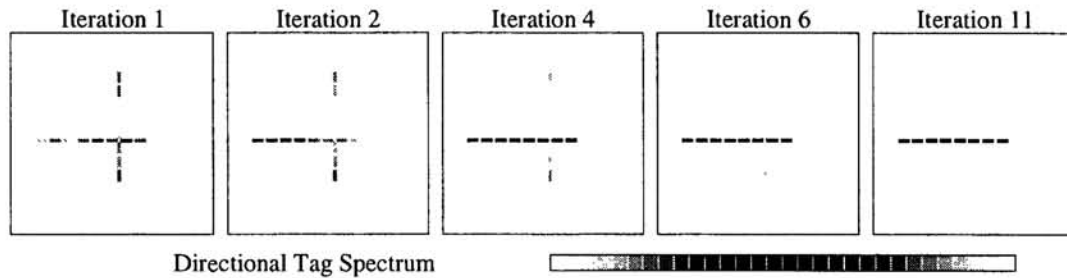

FIGURE 3. The state of MAGIC as processing proceeds for an image composed of a pair of lines made out of horizontal and vertical line segments. The coloring of a segment represents the directional tag. The segments belonging to a line are randomly tagged initially; over processing iterations, these tags are brought into alignment.

pattern over the feature units; that is, the hidden units must encode and propagate information back to the feature units that is sufficient to specify the feature activities (if the feature activities weren't clamped). With this training criterion, MAGIC becomes a type of autoencoder. The key property of MAGIC is that it can assign a feature configuration the same tag only if it learns to encode the configuration. If an arrangement is not encoded, there will be no force to align the feature tags. Further, fixed weak inhibitory connections between every pair of feature units serve to spread the tags apart if the force to align them is not strong enough.

Note that this training paradigm does not require a teacher to tag features as belonging to one part or another. MAGIC will try to tag all features as belonging to the same part, but it is able to do so only for configurations of features that it is able to encode. Consequently, highly regular and recurring configurations will be grouped together, and irregular configurations will be pulled apart. The strength of grouping will be proportional to the degree of regularity.

## 6  SIMULATION EXPERIMENTS

To illustrate the behavior of the model, I show a simple simulation in which MAGIC is trained on pairs of lines, one vertical and one horizontal. Each line is made up of 6 colinear line segments. The segments are primitive input features of the model. The two lines may appear in different positions relative to one another. Hence, the strongest regularity is in the segments that make up a line, not the junction between the lines. When trained with two hidden units, MAGIC has sufficient resources to encode the structure within each line, but not the relationships among the lines; because this structure is not encoded, the features of the two lines are not assigned the same tags (Figure 3).

Although each "part" is made up of features having a uniform orientation and in a colinear arrangement, the composition and structure of the parts is immaterial; MAGIC's performance depends only on the regularity of the configurations. In the next set of simulations, MAGIC discovers regularities of a more arbitrary nature.

### 6.1  MODELING HUMAN LEARNING OF PART-WHOLE RELATIONS

Schyns and Murphy (1992) studied the ontogeny of part-whole relationships by training human subjects on a novel class of objects and then examining how the subjects decomposed the objects into their parts. I briefly describe their experiment, followed by a simulation that accounts for their results.

In the first phase of the experiment, subjects were shown 3-D gray level "martian rocks" on a CRT screen. The rocks were constructed by deforming a sphere, resulting in various bumps or protrusions. Subjects watched the rocks rotating on the screen, allowing them to view the rock from all sides. Subjects were shown six instances, all of which were labeled "M1 rocks" and were then tested to determine whether they could distinguish M1

rocks from other rocks. Subjects continued training until they performed correctly on this task. Every M1 rock was divided into octants; the protrusions on seven of the octants were generated randomly, and the protrusions on the last octant were the same for all M1 rocks. Two groups of subjects were studied. The A group saw M1 rocks all having part A, the B group saw M1 rocks all having part B. Following training, subjects were asked to delineate the parts they thought were important on various exemplars. Subjects selected the target part from the category on which they were trained 93% of the time, and the *alternative target*—the target from the other category—only 8% of the time, indicating that the learning task made a part dramatically more salient.

To model this phase of the experiment, I generated two dimensional contours of the same flavor as Schyns and Murphy's martian rocks (Figure 4). Each rock—call it a "venusian rock" for distinction—can be divided into four quadrants or parts. Two groups of venusian rocks were generated. Rocks of category A all contained part A (left panel, Figure 4), rocks of category B contained part B (center panel, Figure 4). One network was trained on six exemplars of category A rocks, another network was trained on six exemplars of category B rocks. Then, with learning turned off, both networks were tested on five presentations each of twelve new exemplars, six each of categories A and B.

Just as the human subjects were instructed to delineate parts, we must ask MAGIC to do the same. One approach would be to run the model with a test stimulus and, once it settles, select all features having directional tags clustered tightly together as belonging to the same part. However, this requires specifying and tuning a clustering procedure. To avoid this additional step, I simply compared how tightly clustered were the tags of the target part relative to those of the alternative target. I used a directional variance measure that yields a value of 0 if all tags are identical and 1 if the tags are distributed uniformly over the directional spectrum. By this measure, the variance was .30 for the target part and .68 for the alternative target ($F(1,118) = 322.0$, $p < .001$), indicating that the grouping of features of the target part was significantly stronger. This replicates, at least qualitatively, the finding of Schyns and Murphy.

In a second phase of Schyns and Murphy's experiment, subjects were trained on category C rocks, which were formed by adjoining parts A and B and generating the remaining six octants at random. Following training, subjects were again asked to delineate parts. All subjects delineated A and B as distinct parts. In contrast, a naive group of subjects who were trained on category C alone always grouped A and B together as a single part.

To model this phase, I generated six category C venusian rocks that had both parts A and B (right panel, Figure 4). The versions of MAGIC that had been trained on category A and B rocks alone were now trained on category C rocks. As a control condition, a third version of MAGIC was trained from scratch on category C rocks alone. I compared the tightness of clustering of the combined A-B part for the first two nets to the third. Using the same variance measure as above, the nets that first received training on parts A and B alone yielded a variance of .57, and the net that was only trained on the combined A-B part yielded a variance of .47 ($F(1,88) = 7.02$, $p < .02$). One cannot directly compare the variance of the A-B part to that of the A and B parts alone, because the measure is structured such that parts with more features always yield larger variances. However, one can compare the two conditions using the relative variance of the combined A-B part to the A

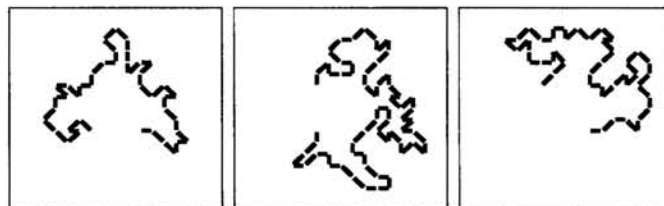

**FIGURE 4. Three examples of the martian rock stimuli used to train MAGIC. From left to right, the rocks are of categories A, B, and C. The lighter regions are the contours that define rocks of a given category.**

and B parts alone. This yielded the same outcome as before (.21 for the first two nets, .12 for the third net, $F(1,88) = 5.80$, $p < .02$). Thus, MAGIC is also able to account for the effects of prior learning on part ontogeny.

## 7  CONCLUSIONS

The regularity principle proposed in this work seems consistent with the homogeneity principle proposed earlier by Schyns and Murphy (1991, 1992). Indeed, MAGIC is able to model Schyns and Murphy's data using an unsupervised training paradigm, although Schyns and Murphy framed their experiment as a classification task.

This work is but a start at modeling the development of part-whole hierarchies based on perceptual experience. MAGIC requires further elaboration, and I am somewhat skeptical that it is sufficiently powerful in its present form to be pushed much further. The main issue restricting it is the representation of input features. The oriented-line-segment features are certainly too primitive and inflexible a representation. For example, MAGIC could not be trained to recognize the lid and shell of Figure 2 because it encodes the orientation of the features with respect to the image plane, not with respect to one another. Minimally, the representation requires some version of scale and rotation invariance.

Perhaps the most interesting computational issue raised by MAGIC is how the pattern of feature tags is mapped into an explicit part-whole decomposition. This involves clustering together the similar tags as a unit, or possibly selecting all tags in a given range. To do so requires specification of additional parameters that are external to the model (e.g., how tight the cluster should be, how broad the range should be, around what tag direction it should be centered). These parameters are deeply related to attentional issues, and a current direction of research is to explore this relationship.

## 8  ACKNOWLEDGEMENTS

This research was supported by NSF PYI award IRI-9058450 and grant 97-18 from the McDonnell-Pew Program in Cognitive Neuroscience.

## 9  REFERENCES

Becker, S. (1995). JPMAX: Learning to recognize moving objects as a model-fitting problem. In G. Tesauro, D. S. Touretzky, & T. K. Leen (Eds), *Advances in Neural Information Processing Systems 7* (pp. 933-940). Cambridge, MA: MIT Press.

Eckhorn, R., Bauer, R., Jordan, W., Brosch, M., Kruse, W., Munk, M., & Reitboek, H. J. (1988). Coherent oscillations: A mechanism of feature linking in the visual cortex? *Biological Cybernetics, 60*, 121–130.

Gray, C. M., Koenig, P., Engel, A. K., & Singer, W. (1989). Oscillatory responses in cat visual cortex exhibit intercolumnar synchronization which reflects global stimulus properties. *Nature* (London), *338*, 334–337.

Mozer, M. C., Zemel, R. S., Behrmann, M., & Williams, C. K. I. (1992). Learning to segment images using dynamic feature binding. *Neural Computation, 4*, 650–666.

Plaut, D. C., Nowlan, S., & Hinton, G. E. (1986). *Experiments on learning by back propagation* (Technical report CMU-CS-86-126). Pittsburgh, PA: Carnegie-Mellon University, Department of Computer Science.

Pollack, J. B. (1988). Recursive auto-associative memory: Devising compositional distributed representations. In *Proceedings of the Tenth Annual Conference of the Cognitive Science Society* (pp. 33–39). Hillsdale, NJ: Erlbaum.

Schyns, P. G., & Murphy, G. L. (1992). The ontogeny of units in object categories. *In Proceedings of the Fourteenth Annual Conference of the Cognitive Science Society* (pp. 197–202). Hillsdale, NJ: Erlbaum.

Schyns, P. G., & Murphy, G. L. (1993). The ontogeny of transformable part representations in object concepts. In *Proceedings of the Fifteenth Annual Conference of the Cognitive Science Society* (pp. 917–922). Hillsdale, NJ: Erlbaum.

Smolensky, P. (1990). Tensor product variable binding and the representation of symbolic structures in connectionist networks. *Artificial Intelligence, 46*, 159–216.

Tenenbaum, J. B. (1994). Functional parts. In A. Ram & K. Eiselt (Eds.), *Proceedings of the Sixteenth Annual Conference of the Cognitive Science Society* (pp. 864–869). Hillsdale, NJ: Erlbaum.

von der Malsburg, C. (1981). *The correlation theory of brain function* (Internal Report 81-2). Goettingen: Department of Neurobiology, Max Planck Institute for Biophysical Chemistry.

Zemel, R. S., Williams, C. K. I., & Mozer, M. C. (1995). Lending direction to neural networks. *Neural Networks, 8*, 503–512.